# Adaptive Regularization of Weight Vectors

**Koby Crammer**
Department of
Electrical Enginering
The Technion
Haifa, 32000 Israel
koby@ee.technion.ac.il

**Alex Kulesza**
Department of Computer
and Information Science
University of Pennsylvania
Philadelphia, PA 19104
kulesza@cis.upenn.edu

**Mark Dredze**
Human Language Tech.
Center of Excellence
Johns Hopkins University
Baltimore, MD 21211
mdredze@cs.jhu.edu

## Abstract

We present AROW, a new online learning algorithm that combines several useful properties: large margin training, confidence weighting, and the capacity to handle non-separable data. AROW performs adaptive regularization of the prediction function upon seeing each new instance, allowing it to perform especially well in the presence of label noise. We derive a mistake bound, similar in form to the second order perceptron bound, that does not assume separability. We also relate our algorithm to recent confidence-weighted online learning techniques and show empirically that AROW achieves state-of-the-art performance and notable robustness in the case of non-separable data.

## 1  Introduction

Online learning algorithms are fast, simple, make few statistical assumptions, and perform well in a wide variety of settings. Recent work has shown that parameter confidence information can be effectively used to guide online learning [2]. Confidence weighted (CW) learning, for example, maintains a Gaussian distribution over linear classifier hypotheses and uses it to control the direction and scale of parameter updates [6]. In addition to formal guarantees in the mistake-bound model [11], CW learning has achieved state-of-the-art performance on many tasks. However, the strict update criterion used by CW learning is very aggressive and can over-fit [5]. Approximate solutions can be used to regularize the update and improve results; however, current analyses of CW learning still assume that the data are separable. It is not immediately clear how to relax this assumption.

In this paper we present a new online learning algorithm for binary classification that combines several attractive properties: large margin training, confidence weighting, and the capacity to handle non-separable data. The key to our approach is the adaptive regularization of the prediction function upon seeing each new instance, so we call this algorithm Adaptive Regularization of Weights (AROW). Because it adjusts its regularization for each example, AROW is robust to sudden changes in the classification function due to label noise. We derive a mistake bound, similar in form to the second order perceptron bound, that does not assume separability. We also provide empirical results demonstrating that AROW is competitive with state-of-the-art methods and improves upon them significantly in the presence of label noise.

## 2  Confidence Weighted Online Learning of Linear Classifiers

Online algorithms operate in rounds. In round $t$ the algorithm receives an instance $\boldsymbol{x}_t \in \mathbb{R}^d$ and applies its current prediction rule to make a prediction $\hat{y}_t \in \mathcal{Y}$. It then receives the true

label $y_t \in \mathcal{Y}$ and suffers a loss $\ell(y_t, \hat{y}_t)$. For binary classification we have $\mathcal{Y} = \{-1, +1\}$ and use the zero-one loss $\ell_{01}(y_t, \hat{y}_t) = 0$ if $y_t = \hat{y}_t$ and 1 otherwise. Finally, the algorithm updates its prediction rule using $(\boldsymbol{x}_t, y_t)$ and proceeds to the next round. In this work we consider linear prediction rules parameterized by a weight vector $\boldsymbol{w}$: $\hat{y} = h_{\boldsymbol{w}}(\boldsymbol{x}) = \text{sign}(\boldsymbol{w} \cdot \boldsymbol{x})$.

Recently Dredze, Crammer and Pereira [6, 5] proposed an algorithmic framework for online learning of binary classification tasks called confidence weighted (CW) learning. CW learning captures the notion of confidence in a linear classifier by maintaining a Gaussian distribution over the weights with mean $\boldsymbol{\mu} \in \mathbb{R}^d$ and covariance matrix $\Sigma \in \mathbb{R}^{d \times d}$. The values $\mu_p$ and $\Sigma_{p,p}$, respectively, encode the learner's knowledge of and confidence in the weight for feature $p$: the smaller $\Sigma_{p,p}$, the more confidence the learner has in the mean weight value $\mu_p$. Covariance terms $\Sigma_{p,q}$ capture interactions between weights.

Conceptually, to classify an instance $\boldsymbol{x}$, a CW classifier draws a parameter vector $\boldsymbol{w} \sim \mathcal{N}(\boldsymbol{\mu}, \Sigma)$ and predicts the label according to $\text{sign}(\boldsymbol{w} \cdot \boldsymbol{x})$. In practice, however, it can be easier to simply use the average weight vector $\text{E}[\boldsymbol{w}] = \boldsymbol{\mu}$ to make predictions. This is similar to the approach taken by Bayes point machines [9], where a single weight vector is used to approximate a distribution. Furthermore, for binary classification, the prediction given by the mean weight vector turns out to be Bayes optimal.

CW classifiers are trained according to a passive-aggressive rule [3] that adjusts the distribution at each round to ensure that the probability of a correct prediction is at least $\eta \in (0.5, 1]$. This yields the update constraint $\text{Pr}[y_t(\boldsymbol{w} \cdot \boldsymbol{x}_t) \geq 0] \geq \eta$. Subject to this constraint, the algorithm makes the smallest possible change to the hypothesis weight distribution as measured using the KL divergence. This implies the following optimization problem for each round $t$:

$$(\boldsymbol{\mu}_t, \Sigma_t) = \min_{\boldsymbol{\mu}, \Sigma} \text{D}_{\text{KL}}\left(\mathcal{N}(\boldsymbol{\mu}, \Sigma) \parallel \mathcal{N}(\boldsymbol{\mu}_{t-1}, \Sigma_{t-1})\right)$$

$$\text{s.t. } \text{Pr}_{\boldsymbol{w} \sim \mathcal{N}(\boldsymbol{\mu}, \Sigma)}[y_t(\boldsymbol{w} \cdot \boldsymbol{x}_t) \geq 0] \geq \eta$$

Confidence-weighted algorithms have been shown to perform well in practice [5, 6], but they suffer from several problems. First, the update is quite aggressive, forcing the probability of predicting each example correctly to be at least $\eta > 1/2$ regardless of the cost to the objective. This may cause severe over-fitting when labels are noisy; indeed, current analyses of the CW algorithm [5] assume that the data are linearly separable. Second, they are designed for classification, and it is not clear how to extend them to alternative settings such as regression. This is in part because the constraint is written in discrete terms where the prediction is either correct or not.

We deal with both of these issues, coping more effectively with label noise and generalizing the advantages of CW learning in an extensible way.

## 3   Adaptive Regularization Of Weights

We identify two important properties of the CW update rule that contribute to its good performance but also make it sensitive to label noise. First, the mean parameters $\boldsymbol{\mu}$ are guaranteed to correctly classify the current training example with margin following each update. This is because the probability constraint $\text{Pr}[y_t(\boldsymbol{w} \cdot \boldsymbol{x}_t) \geq 0] \geq \eta$ can be written explicitly as $y_t(\boldsymbol{\mu} \cdot \boldsymbol{x}_t) \geq \phi\sqrt{\boldsymbol{x}_t^\top \Sigma \boldsymbol{x}_t}$, where $\phi > 0$ is a positive constant related to $\eta$. This aggressiveness yields rapid learning, but given an incorrectly labeled example, it can also force the learner to make a drastic and incorrect change to its parameters. Second, confidence, as measured by the inverse eigenvalues of $\Sigma$, increases monotonically with every update. While it is intuitive that our confidence should grow as we see more data, this also means that even incorrectly labeled examples causing wild parameter swings result in artificially increased confidence.

In order to maintain the positives but reduce the negatives of these two properties, we isolate and soften them. As in CW learning, we maintain a Gaussian distribution over weight vectors with mean $\boldsymbol{\mu}$ and covariance $\Sigma$; however, we recast the above characteristics of the CW constraint as regularizers, minimizing the following unconstrained objective on

each round:

$$\mathcal{C}\left(\boldsymbol{\mu},\Sigma\right) = \mathrm{D_{KL}}\left(\mathcal{N}\left(\boldsymbol{\mu},\Sigma\right)\|\mathcal{N}\left(\boldsymbol{\mu}_{t-1},\Sigma_{t-1}\right)\right) + \lambda_1\ell_{\mathrm{h}^2}\left(y_t, \boldsymbol{\mu}\cdot\boldsymbol{x}_t\right) + \lambda_2\boldsymbol{x}_t^\top\Sigma\boldsymbol{x}_t\ , \qquad (1)$$

where $\ell_{\mathrm{h}^2}\left(y_t,\boldsymbol{\mu}\cdot\boldsymbol{x}_t\right) = \left(\max\{0, 1-y_t(\boldsymbol{\mu}\cdot\boldsymbol{x}_t)\}\right)^2$ is the squared-hinge loss suffered using the weight vector $\boldsymbol{\mu}$ to predict the output for input $\boldsymbol{x}_t$ when the true output is $y_t$. $\lambda_1,\lambda_2 \geq 0$ are two tradeoff hyperparameters. For simplicity and compactness of notation, in the following we will assume that $\lambda_1 = \lambda_2 = 1/(2r)$ for some $r > 0$.

The objective balances three desires. First, the parameters should not change radically on each round, since the current parameters contain information about previous examples (first term). Second, the new mean parameters should predict the current example with low loss (second term). Finally, as we see more examples, our confidence in the parameters should generally grow (third term).

Note that this objective is not simply the dualization of the CW constraint, but a new formulation inspired by the properties discussed above. Since the loss term depends on $\boldsymbol{\mu}$ only via the inner-product $\boldsymbol{\mu}\cdot\boldsymbol{x}_t$, we are able to prove a representer theorem (Sec. 4). While we use the squared-hinge loss for classification, different loss functions, as long as they are convex and differentiable in $\boldsymbol{\mu}$, yield algorithms for different settings.[1]

To solve the optimization in (1), we begin by writing the KL explicitly:

$$\mathcal{C}\left(\boldsymbol{\mu},\Sigma\right) = \frac{1}{2}\log\left(\frac{\det\Sigma_{t-1}}{\det\Sigma}\right) + \frac{1}{2}\mathrm{Tr}\left(\Sigma_{t-1}^{-1}\Sigma\right) + \frac{1}{2}\left(\boldsymbol{\mu}_{t-1}-\boldsymbol{\mu}\right)^\top\Sigma_{t-1}^{-1}\left(\boldsymbol{\mu}_{t-1}-\boldsymbol{\mu}\right) - \frac{d}{2}$$

$$+ \frac{1}{2r}\ell_{\mathrm{h}^2}\left(y_t,\boldsymbol{\mu}\cdot\boldsymbol{x}_t\right) + \frac{1}{2r}\boldsymbol{x}_t^\top\Sigma\boldsymbol{x}_t \quad (2)$$

We can decompose the result into two terms: $\mathcal{C}_1(\boldsymbol{\mu})$, depending only on $\boldsymbol{\mu}$, and $\mathcal{C}_2(\Sigma)$, depending only on $\Sigma$. The updates to $\boldsymbol{\mu}$ and $\Sigma$ can therefore be performed independently. The squared-hinge loss yields a conservative (or passive) update for $\boldsymbol{\mu}$ in which the mean parameters change only when the margin is too small, and we follow CW learning by enforcing a correspondingly conservative update for the confidence parameter $\Sigma$, updating it only when $\boldsymbol{\mu}$ changes. This results in fewer updates and is easier to analyze. Our update thus proceeds in two stages.

1. Update the mean parameters: $\qquad\qquad\qquad\qquad\qquad \boldsymbol{\mu}_t = \arg\min_{\boldsymbol{\mu}}\mathcal{C}_1\left(\boldsymbol{\mu}\right) \qquad (3)$

2. If $\boldsymbol{\mu}_t \neq \boldsymbol{\mu}_{t-1}$, update the confidence parameters: $\qquad \Sigma_t = \arg\min_{\Sigma}\mathcal{C}_2\left(\Sigma\right) \qquad (4)$

We now develop the update equations for (3) and (4) explicitly, starting with the former. Taking the derivative of $\mathcal{C}\left(\boldsymbol{\mu},\Sigma\right)$ with respect to $\boldsymbol{\mu}$ and setting it to zero, we get

$$\boldsymbol{\mu}_t = \boldsymbol{\mu}_{t-1} - \frac{1}{2r}\left[\frac{d}{dz}\ell_{\mathrm{h}^2}\left(y_t, z\right)|_{z=\boldsymbol{\mu}_t\cdot\boldsymbol{x}_t}\right]\Sigma_{t-1}\boldsymbol{x}_t\ , \qquad (5)$$

assuming $\Sigma_{t-1}$ is non-singular. Substituting the derivative of the squared-hinge loss in (5) and assuming $1 - y_t\left(\boldsymbol{\mu}_t\cdot\boldsymbol{x}_t\right) \geq 0$, we get

$$\boldsymbol{\mu}_t = \boldsymbol{\mu}_{t-1} + \frac{y_t}{r}\left(1 - y_t\left(\boldsymbol{\mu}_t\cdot\boldsymbol{x}_t\right)\right)\Sigma_{t-1}\boldsymbol{x}_t\ . \qquad (6)$$

We solve for $\boldsymbol{\mu}_t$ by taking the dot product of each side of the equality with $\boldsymbol{x}_t$ and substituting back in (6) to obtain the rule

$$\boldsymbol{\mu}_t = \boldsymbol{\mu}_{t-1} + \frac{\max\left(0, 1 - y_t\boldsymbol{x}_t^\top\boldsymbol{\mu}_{t-1}\right)}{\boldsymbol{x}_t^\top\Sigma_{t-1}\boldsymbol{x}_t + r}\Sigma_{t-1}y_t\boldsymbol{x}_t\ . \qquad (7)$$

It can be easily verified that (7) satisfies our assumption that $1 - y_t\left(\boldsymbol{\mu}_t\cdot\boldsymbol{x}_t\right) \geq 0$.

**Input parameters** $r$
**Initialize** $\boldsymbol{\mu}_0 = \mathbf{0}$, $\Sigma_0 = I$,
**For** $t = 1, \ldots, T$
  - Receive a training example $\boldsymbol{x}_t \in \mathbb{R}^d$
  - Compute margin and confidence $m_t = \boldsymbol{\mu}_{t-1} \cdot \boldsymbol{x}_t \quad v_t = \boldsymbol{x}_t^\top \Sigma_{t-1} \boldsymbol{x}_t$
  - Receive true label $y_t$, and suffer loss $\ell_t = 1$ if $\text{sign}(m_t) \neq y_t$
  - If $m_t y_t < 1$, update using eqs. (7) & (9):

$$\boldsymbol{\mu}_t = \boldsymbol{\mu}_{t-1} + \alpha_t \Sigma_{t-1} y_t \boldsymbol{x}_t \qquad\qquad \Sigma_t = \Sigma_{t-1} - \beta_t \Sigma_{t-1} \boldsymbol{x}_t \boldsymbol{x}_t^\top \Sigma_{t-1}$$

$$\beta_t = \frac{1}{\boldsymbol{x}_t^\top \Sigma_{t-1} \boldsymbol{x}_t + r} \qquad\qquad \alpha_t = \max\left(0, 1 - y_t \boldsymbol{x}_t^\top \boldsymbol{\mu}_{t-1}\right) \beta_t$$

**Output:** Weight vector $\boldsymbol{\mu}_T$ and confidence $\Sigma_T$.

---

Figure 1: The AROW algorithm for online binary classification.

The update for the confidence parameters is made only if $\boldsymbol{\mu}_t \neq \boldsymbol{\mu}_{t-1}$, that is, if $1 > y_t \boldsymbol{x}_t^\top \boldsymbol{\mu}_{t-1}$. In this case, we compute the update of the confidence parameters by setting the derivative of $\mathcal{C}(\boldsymbol{\mu}, \Sigma)$ with respect to $\Sigma$ to zero:

$$\Sigma_t^{-1} = \Sigma_{t-1}^{-1} + \frac{\boldsymbol{x}_t \boldsymbol{x}_t^\top}{r} \tag{8}$$

Using the Woodbury identity we can also rewrite the update for $\Sigma$ in non-inverted form:

$$\Sigma_t = \Sigma_{t-1} - \frac{\Sigma_{t-1} \boldsymbol{x}_t \boldsymbol{x}_t^\top \Sigma_{t-1}}{r + \boldsymbol{x}_t^\top \Sigma_{t-1} \boldsymbol{x}_t} \tag{9}$$

Note that it follows directly from (8) and (9) that the eigenvalues of the confidence parameters are monotonically decreasing: $\Sigma_t \preceq \Sigma_{t-1}$; $\Sigma_t^{-1} \succeq \Sigma_{t-1}^{-1}$ . Pseudocode for AROW appears in Fig. 1.

## 4 Analysis

We first show that AROW can be kernelized by stating the following representer theorem.

**Lemma 1 (Representer Theorem)** *Assume that $\Sigma_0 = I$ and $\boldsymbol{\mu}_0 = \mathbf{0}$. The mean parameters $\boldsymbol{\mu}_t$ and confidence parameters $\Sigma_t$ produced by updating via (7) and (9) can be written as linear combinations of the input vectors (resp. outer products of the input vectors with themselves) with coefficients depending only on inner-products of input vectors.*

**Proof sketch:** By induction. The base case follows from the definitions of $\boldsymbol{\mu}_0$ and $\Sigma_0$, and the induction step follows algebraically from the update rules (7) and (9).

We now prove a mistake bound for AROW. Denote by $\mathcal{M}$ ($M = |\mathcal{M}|$) the set of example indices for which the algorithm makes a mistake, $y_t (\boldsymbol{\mu}_{t-1} \cdot \boldsymbol{x}_t) \leq 0$, and by $\mathcal{U}$ ($U = |\mathcal{U}|$) the set of example indices for which there is an update but not a mistake, $0 < y_t (\boldsymbol{\mu}_t \cdot \boldsymbol{x}_t) \leq 1$. Other examples do not affect the behavior of the algorithm and can be ignored. Let $\mathbf{X}_{\mathcal{M}} = \sum_{t \in \mathcal{M}} \boldsymbol{x}_i \boldsymbol{x}_i^\top$, $\mathbf{X}_{\mathcal{U}} = \sum_{t \in \mathcal{U}} \boldsymbol{x}_i \boldsymbol{x}_i^\top$ and $\mathbf{X}_{\mathcal{A}} = \mathbf{X}_{\mathcal{M}} + \mathbf{X}_{\mathcal{U}}$.

**Theorem 2** *For any reference weight vector $\boldsymbol{u} \in \mathbb{R}^d$, the number of mistakes made by AROW (Fig. 1) is upper bounded by*

$$M \leq \sqrt{r \|\boldsymbol{u}\|^2 + \boldsymbol{u}^\top \mathbf{X}_{\mathcal{A}} \boldsymbol{u}} \sqrt{\log\left(\det\left(I + \frac{1}{r} \mathbf{X}_{\mathcal{A}}\right)\right) + U} + \sum_{t \in \mathcal{M} \cup \mathcal{U}} g_t - U \;, \tag{10}$$

*where $g_t = \max\left(0, 1 - y_t \boldsymbol{u}^\top \boldsymbol{x}_t\right)$.*

The proof depends on two lemmas; we omit the proof of the first for lack of space.

**Lemma 3** *Let* $\ell_t = \max\left(0, 1 - y_t \boldsymbol{\mu}_{t-1}^\top \boldsymbol{x}_t\right)$ *and* $\chi_t = \boldsymbol{x}_t^\top \Sigma_{t-1} \boldsymbol{x}_t$. *Then, for every* $t \in \mathcal{M} \cup \mathcal{U}$,

$$\boldsymbol{u}^\top \Sigma_t^{-1} \boldsymbol{\mu}_t = \boldsymbol{u}^\top \Sigma_{t-1}^{-1} \boldsymbol{\mu}_{t-1} + \frac{y_t \boldsymbol{u}^\top \boldsymbol{x}_t}{r}$$

$$\boldsymbol{\mu}_t^\top \Sigma_t^{-1} \boldsymbol{\mu}_t = \boldsymbol{\mu}_{t-1}^\top \Sigma_{t-1}^{-1} \boldsymbol{\mu}_{t-1} + \frac{\chi_t + r - \ell_t^2 r}{r(\chi_t + r)}$$

**Lemma 4** *Let $T$ be the number of rounds. Then*

$$\sum_t \frac{\chi_t r}{r(\chi_t + r)} \leq \log\left(\det\left(\Sigma_{T+1}^{-1}\right)\right) \ .$$

**Proof:** We compute the following quantity:

$$\boldsymbol{x}_t^\top \Sigma_t \boldsymbol{x}_t^\top = \boldsymbol{x}_t^\top \left(\Sigma_{t-1} - \beta_t \Sigma_{t-1} \boldsymbol{x}_t \boldsymbol{x}_t^\top \Sigma_{t-1}\right) \boldsymbol{x}_t = \chi_t - \frac{\chi_t^2}{\chi_t + r} = \frac{\chi_t r}{\chi_t + r} \ .$$

Using Lemma D.1 from [2] we have that

$$\frac{1}{r} \boldsymbol{x}_t^\top \Sigma_t \boldsymbol{x}_t^\top = 1 - \frac{\det\left(\Sigma_{t-1}^{-1}\right)}{\det\left(\Sigma_t^{-1}\right)} \ . \tag{11}$$

Combining, we get

$$\sum_t \frac{\chi_t r}{r(\chi_t + r)} = \sum_t \left(1 - \frac{\det\left(\Sigma_{t-1}^{-1}\right)}{\det\left(\Sigma_t^{-1}\right)}\right) \leq -\sum_t \log\left(\frac{\det\left(\Sigma_{t-1}^{-1}\right)}{\det\left(\Sigma_t^{-1}\right)}\right) \leq \log\left(\det\left(\Sigma_{T+1}^{-1}\right)\right) \ .$$

We now prove Theorem 2.

**Proof:** We iterate the first equality of Lemma 3 to get

$$\boldsymbol{u}^\top \Sigma_T^{-1} \boldsymbol{\mu}_T = \sum_{t \in \mathcal{M} \cup \mathcal{U}} \frac{y_t \boldsymbol{u}^\top \boldsymbol{x}_t}{r} \geq \sum_{t \in \mathcal{M} \cup \mathcal{U}} \frac{1 - g_t}{r} = \frac{M + U}{r} - \frac{1}{r} \sum_{t \in \mathcal{M} \cup \mathcal{U}} g_t \ . \tag{12}$$

We iterate the second equality to get

$$\boldsymbol{\mu}_T^\top \Sigma_T^{-1} \boldsymbol{\mu}_T = \sum_{t \in \mathcal{M} \cup \mathcal{U}} \frac{\chi_t + r - \ell_t^2 r}{r(\chi_t + r)} = \sum_{t \in \mathcal{M} \cup \mathcal{U}} \frac{\chi_t}{r(\chi_t + r)} + \sum_{t \in \mathcal{M} \cup \mathcal{U}} \frac{1 - \ell_t^2}{\chi_t + r} \ . \tag{13}$$

Using Lemma 4 we have that the first term of (13) is upper bounded by $\frac{1}{r} \log\left(\det\left(\Sigma_T^{-1}\right)\right)$. For the second term in (13) we consider two cases. First, if a mistake occurred on example $t$, then we have that $y_t\left(\boldsymbol{x}_t \cdot \boldsymbol{\mu}_{t-1}\right) \leq 0$ and $\ell_t \geq 1$, so $1 - \ell_t^2 \leq 0$. Second, if an the algorithm made an update (but no mistake) on example $t$, then $0 < y_t\left(\boldsymbol{x}_t \cdot \boldsymbol{\mu}_{t-1}\right) \leq 1$ and $\ell_t \geq 0$, thus $1 - \ell_t^2 \leq 1$. We therefore have

$$\sum_{t \in \mathcal{M} \cup \mathcal{U}} \frac{1 - \ell_t^2}{\chi_t + r} \leq \sum_{t \in \mathcal{M}} \frac{0}{\chi_t + r} + \sum_{t \in \mathcal{U}} \frac{1}{\chi_t + r} = \sum_{t \in \mathcal{U}} \frac{1}{\chi_t + r} \ . \tag{14}$$

Combining and plugging into the Cauchy-Schwarz inequality

$$\boldsymbol{u}^\top \Sigma_T^{-1} \boldsymbol{\mu}_T \leq \sqrt{\boldsymbol{u}^\top \Sigma_T^{-1} \boldsymbol{u}} \sqrt{\boldsymbol{\mu}_T^\top \Sigma_T^{-1} \boldsymbol{\mu}_T} \ ,$$

we get

$$\frac{M + U}{r} - \frac{1}{r} \sum_{t \in \mathcal{M} \cup \mathcal{U}} g_t \leq \sqrt{\boldsymbol{u}^\top \Sigma_T^{-1} \boldsymbol{u}} \sqrt{\frac{1}{r} \log\left(\det\left(\Sigma_T^{-1}\right)\right) + \sum_{t \in \mathcal{U}} \frac{1}{\chi_t + r}} \ . \tag{15}$$

Rearranging the terms and using the fact that $\chi_t \geq 0$ yields

$$M \leq \sqrt{r} \sqrt{\boldsymbol{u}^\top \Sigma_T^{-1} \boldsymbol{u}} \sqrt{\log\left(\det\left(\Sigma_T^{-1}\right)\right) + U} + \sum_{t \in \mathcal{M} \cup \mathcal{U}} g_t - U \ .$$

By definition,

$$\Sigma_T^{-1} = I + \frac{1}{r} \sum_{t \in \mathcal{M} \cup \mathcal{U}} \boldsymbol{x}_i \boldsymbol{x}_i^\top = I + \frac{1}{r} \mathbf{X}_{\mathcal{A}} \ ,$$

so substituting and simplifying completes the proof:

$$M \leq \sqrt{r} \sqrt{\boldsymbol{u}^\top \left( I + \frac{1}{r}\mathbf{X}_{\mathcal{A}} \right) \boldsymbol{u}} \sqrt{\log\left( \det\left( I + \frac{1}{r}\mathbf{X}_{\mathcal{A}} \right) \right) + U} + \sum_{t \in \mathcal{M} \cup \mathcal{U}} g_t - U$$

$$= \sqrt{r \left\| \boldsymbol{u} \right\|^2 + \boldsymbol{u}^\top \mathbf{X}_{\mathcal{A}} \boldsymbol{u}} \sqrt{\log\left( \det\left( I + \frac{1}{r}\mathbf{X}_{\mathcal{A}} \right) \right) + U} + \sum_{t \in \mathcal{M} \cup \mathcal{U}} g_t - U \ .$$

A few comments are in order. First, the two square-root terms of the bound depend on $r$ in opposite ways: the first is monotonically increasing, while the second is monotonically decreasing. One could expect to optimize the bound by minimizing over $r$. However, the bound also depends on $r$ indirectly via other quantities (e.g. $\mathbf{X}_{\mathcal{A}}$), so there is no direct way to do so. Second, if all the updates are associated with errors, that is, $\mathcal{U} = \emptyset$, then the bound reduces to the bound of the second-order perceptron [2]. In general, however, the bounds are not comparable since each depends on the actual runtime behavior of its algorithm.

## 5   Empirical Evaluation

We evaluate AROW on both synthetic and real data, including several popular datasets for document classification and optical character recognition (OCR). We compare with three baselines: Passive-Aggressive (PA), Second Order Perceptron (SOP)[2] and Confidence-Weighted (CW) learning[3].

Our synthetic data are as in [5], but we invert the labels on 10% of the training examples. (Note that evaluation is still done against the true labels.) Fig. 2(a) shows the online learning curves for both full and diagonalized versions of the algorithms on these noisy data. AROW improves over all competitors, and the full version outperforms the diagonal version. Note that CW-full performs worse than CW-diagonal, as has been observed previously for noisy data.

We selected a variety of document classification datasets popular in the NLP community, summarized as follows. **Amazon**: Product reviews to be classified into domains (e.g., books or music) [6]. We created binary datasets by taking all pairs of the six domains (15 datasets). Feature extraction follows [1] (bigram counts). **20 Newsgroups**: Approximately 20,000 newsgroup messages partitioned across 20 different newsgroups[4]. We binarized the corpus following [6] and used binary bag-of-words features (3 datasets). Each dataset has between 1850 and 1971 instances. **Reuters (RCV1-v2/LYRL2004)**: Over 800,000 manually categorized newswire stories. We created binary classification tasks using pairs of labels following [6] (3 datasets). Details on document preparation and feature extraction are given by [10]. **Sentiment**: Product reviews to be classified as positive or negative. We used each Amazon product review domain as a sentiment classification task (6 datasets). **Spam**: We selected three task A users from the ECML/PKDD Challenge[5], using bag-of-words to classify each email as spam or ham (3 datasets). For OCR data we binarized two well known digit recognition datasets, **MNIST**[6] and **USPS**, into 45 all-pairs problems. We also created ten one vs. all datasets from the MNIST data (100 datasets total).

Each result for the text datasets was averaged over 10-fold cross-validation. The OCR experiments used the standard split into training and test sets. Hyperparameters (including

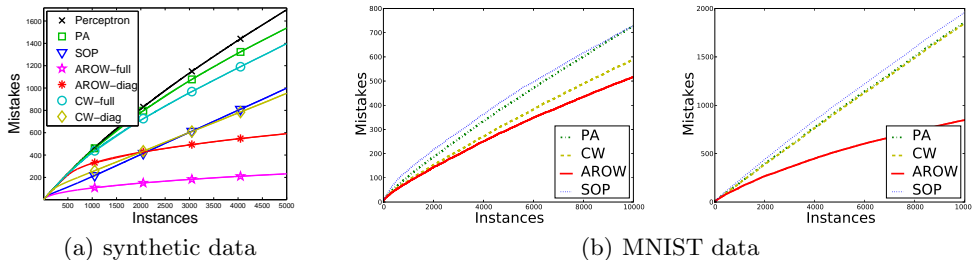

(a) synthetic data          (b) MNIST data

Figure 2: Learning curves for AROW (full/diagonal) and baseline methods. (a) 5k synthetic training examples and 10k test examples (10% noise, 100 runs). (b) MNIST 3 vs. 5 binary classification task for different amounts of label noise (left: 0 noise, right: 10%).

$r$ for AROW) and the number of online iterations (up to 10) were optimized using a single randomized run. We used 2000 instances from each dataset unless otherwise noted above. In order to observe each algorithm's ability to handle non-separable data, we performed each experiment using various levels of artifical label noise, generated by independently flipping each binary label with fixed probability.

## 5.1 Results and Discussion

Our experimental results are summarized in Table 1. AROW outperforms the baselines at all noise levels, but does especially well as noise increases. More detailed results for AROW and CW, the overall best performing baseline, are compared in Fig. 3. AROW and CW are comparable when there is no added noise, with AROW

| Algorithm | Noise level | | | | | |
|---|---|---|---|---|---|---|
| | 0.0 | 0.05 | 0.1 | 0.15 | 0.2 | 0.3 |
| *AROW* | **1.51** | **1.44** | **1.38** | **1.42** | **1.25** | **1.25** |
| *CW* | 1.63 | 1.87 | 1.95 | 2.08 | 2.42 | 2.76 |
| *PA* | 2.95 | 2.83 | 2.78 | 2.61 | 2.33 | 2.08 |
| *SOP* | 3.91 | 3.87 | 3.89 | 3.89 | 4.00 | 3.91 |

Table 1: Mean rank (out of 4, over all datasets) at different noise levels. A rank of 1 indicates that an algorithm outperformed all the others.

winning the majority of the time. As label noise increases (moving across the rows in Fig. 3) AROW holds up remarkably well. In almost every high noise evaluation, AROW improves over CW (as well as the other baselines, not shown). Fig. 2(b) shows the total number of mistakes (w.r.t. noise-free labels) made by each algorithm during training on the MNIST dataset for 0% and 10% noise. Though absolute performance suffers with noise, the gap between AROW and the baselines increases.

To help interpret the results, we classify the algorithms evaluated here according to four characteristics: the use of large margin updates, confidence weighting, a design that accomodates non-separable data, and adaptive per-instance margin (Table 2). While all of these properties can be desirable in different situations, we would like to understand how they interact and achieve high performance while avoiding sensitivity to noise.

Based on the results in Table 1, it is clear that the combination of confidence information and large margin learning is powerful when label noise is low. CW easily outperforms the other baselines in such situations, as it has been shown to do in previous work. However, as noise increases, the separa-

| Algorithm | Large Margin | Conf-idence | Non-Separable | Adaptive Margin |
|---|---|---|---|---|
| *PA* | **Yes** | No | **Yes** | No |
| *SOP* | No | **Yes** | **Yes** | No |
| *CW* | **Yes** | **Yes** | No | **Yes** |
| *AROW* | **Yes** | **Yes** | **Yes** | No |

Table 2: Online algorithm properties overview.

bility assumption inherent in CW appears to reduce its performance considerably.

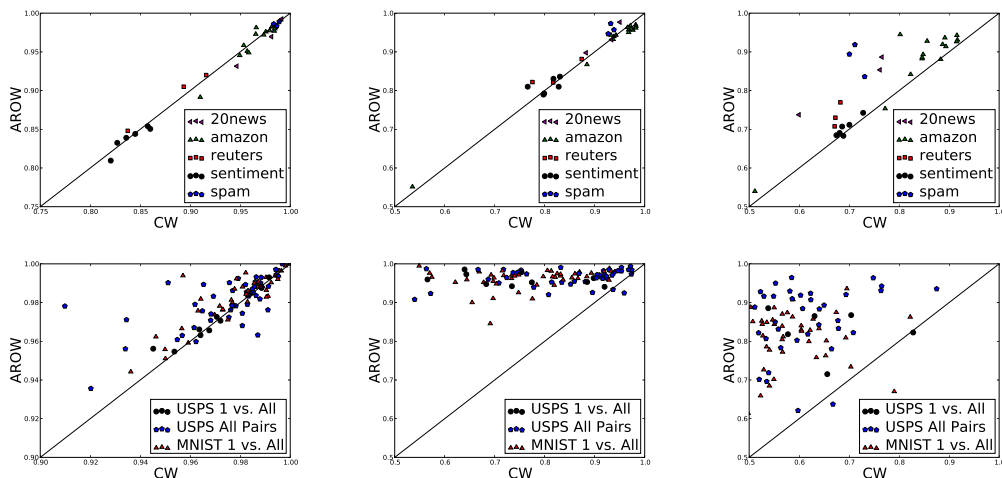

Figure 3: Accuracy on text (top) and OCR (bottom) binary classification. Plots compare performance between AROW and CW, the best performing baseline (Table 1). Markers above the line indicate superior AROW performance and below the line superior CW performance. Label noise increases from left to right: 0%, 10% and 30%. AROW improves relative to CW as noise increases.

AROW, by combining the large margin and confidence weighting of CW with a soft update rule that accomodates non-separable data, matches CW's performance in general while avoiding degradation under noise. AROW lacks the adaptive margin of CW, suggesting that this characteristic is not crucial to achieving strong performance. However, we leave open for future work the possibility that an algorithm with all four properties might have unique advantages.

## 6 Related and Future Work

AROW is most similar to the second order perceptron [2]. The SOP performs the same type of update as AROW, but only when it makes an error. AROW, on the other hand, updates even when its prediction is correct if there is insufficient margin. Confidence weighted (CW) [6, 5] algorithms, by which AROW was inspired, update the mean and confidence parameters simultaneously, while AROW makes a decoupled update and softens the hard constraint of CW. The AROW algorithm can be seen as a variant of the PA-II algorithm from [3] where the regularization is modified according to the data.

Hazan [8] describes a framework for gradient descent algorithms with logarithmic regret in which a quantity similar to $\Sigma_t$ plays an important role. Our algorithm differs in several ways. First, Hazan [8] considers gradient algorithms, while we derive and analyze algorithms that directly solve an optimization problem. Second, we bound the loss directly, not the cumulative sum of regularization and loss. Third, the gradient algorithms perform a projection after making an update (not before) since the norm of the weight vector is kept bounded.

Ongoing work includes the development and analysis of AROW style algorithms for other settings, including a multi-class version following the recent extension of CW to multi-class problems [4]. Our mistake bound can be extended to this case. Applying the ideas behind AROW to regression problems turns out to yield the well known recursive least squares (RLS) algorithm, for which AROW offers new bounds (omitted). Finally, while we used the confidence term $\boldsymbol{x}_t^\top \Sigma \boldsymbol{x}_t$ in (1), we can replace this term with any differentiable, monotonically increasing function $f(\boldsymbol{x}_t^\top \Sigma \boldsymbol{x}_t)$. This generalization may yield additional algorithms.

## Footnotes

[1]It can be shown that the well known recursive least squares (RLS) regression algorithm [7] is a special case of AROW with the squared loss.

[2]For the real world (high dimensional) datasets, we must drop cross-feature confidence terms by projecting onto the set of diagonal matrices, following the approach of [6]. While this may reduce performance, we make the same approximation for all evaluated algorithms.

[3]We use the "variance" version developed in [6].

[4]http://people.csail.mit.edu/jrennie/20Newsgroups/

[5]http://ecmlpkdd2006.org/challenge.html

[6]http://yann.lecun.com/exdb/mnist/index.html

# References

[1] John Blitzer, Mark Dredze, and Fernando Pereira. Biographies, bollywood, boom-boxes and blenders: Domain adaptation for sentiment classification. In *ACL*, 2007.

[2] Nicoló Cesa-Bianchi, Alex Conconi, and Claudio Gentile. A second-order perceptron algorithm. *Siam J. of Comm.*, 34, 2005.

[3] Koby Crammer, Ofer Dekel, Joseph Keshet, Shai Shalev-Shwartz, and Yoram Singer. Online passive-aggressive algorithms. *Journal of Machine Learning Research*, 7:551–585, 2006.

[4] Koby Crammer, Mark Dredze, and Alex Kulesza. Multi-class confidence weighted algorithms. In *Empirical Methods in Natural Language Processing (EMNLP)*, 2009.

[5] Koby Crammer, Mark Dredze, and Fernando Pereira. Exact convex confidence-weighted learning. In *Neural Information Processing Systems (NIPS)*, 2008.

[6] Mark Dredze, Koby Crammer, and Fernando Pereira. Confidence-weighted linear classification. In *International Conference on Machine Learning*, 2008.

[7] Simon Haykin. *Adaptive Filter Theory*. 1996.

[8] Elad Hazan. *Efficient algorithms for online convex optimization and their applications*. PhD thesis, Princeton University, 2006.

[9] Ralf Herbrich, Thore Graepel, and Colin Campbell. Bayes point machines. *Journal of Machine Learning Research (JMLR)*, 1:245–279, 2001.

[10] David D. Lewis, Yiming Yang, Tony G. Rose, and Fan Li. Rcv1: A new benchmark collection for text categorization research. *JMLR*, 5:361–397, 2004.

[11] Nick Littlestone. Learning when irrelevant attributes abound: A new linear-threshold algorithm. *Machine Learning*, 2:285–318, 1988.

